# Analog VLSI Processor Implementing the Continuous Wavelet Transform

**R. Timothy Edwards** and **Gert Cauwenberghs**
Department of Electrical and Computer Engineering
Johns Hopkins University
3400 North Charles Street
Baltimore, MD 21218-2686
{tim,gert}@bach.ece.jhu.edu

## Abstract

We present an integrated analog processor for real-time wavelet decomposition and reconstruction of continuous temporal signals covering the audio frequency range. The processor performs complex harmonic modulation and Gaussian lowpass filtering in 16 parallel channels, each clocked at a different rate, producing a multiresolution mapping on a logarithmic frequency scale. Our implementation uses mixed-mode analog and digital circuits, oversampling techniques, and switched-capacitor filters to achieve a wide linear dynamic range while maintaining compact circuit size and low power consumption. We include experimental results on the processor and characterize its components separately from measurements on a single-channel test chip.

## 1 Introduction

An effective mathematical tool for multiresolution analysis [Kais94], the wavelet transform has found widespread use in various signal processing applications involving characteristic patterns that cover multiple scales of resolution, such as representations of speech and vision. Wavelets offer suitable representations for temporal data that contain pertinent features both in the time and frequency domains; consequently, wavelet decompositions appear to be effective in representing wide-bandwidth signals interfacing with neural systems [Szu92].

The present system performs a continuous wavelet transform on temporal one-dimensional analog signals such as speech, and is in that regard somewhat related to silicon models of the cochlea implementing cochlear transforms [Lyon88], [Liu92], [Watt92], [Lin94]. The multiresolution processor we implemented expands on the architecture developed in [Edwa93], which differs from the other analog auditory processors in the way signal components in each frequency band are encoded. The signal is modulated with the center

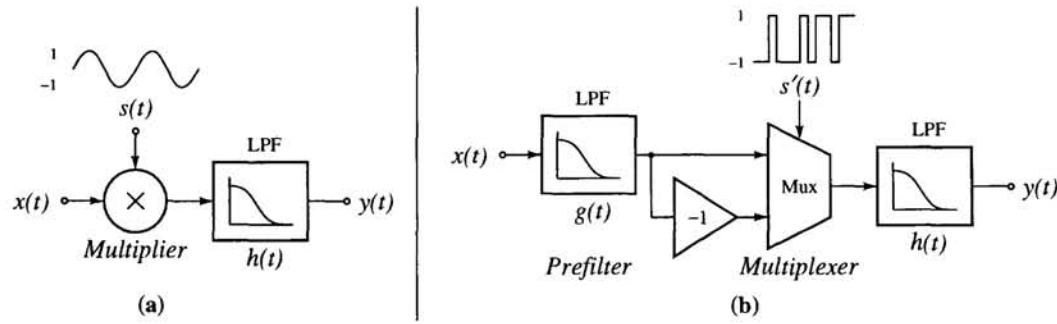

Figure 1: *Demodulation systems, (a) using multiplication, and (b) multiplexing.*

frequency of each channel and subsequently lowpass filtered, translating signal components taken around the center frequency towards zero frequency. In particular, we consider wavelet decomposition and reconstruction of analog continuous-time temporal data with a complex Gaussian kernel according to the following formulae:

$$y_k(t) = \int_{-\infty}^{t} x(\theta) \, \exp\left(j\omega_k\theta - \alpha(\omega_k(t - \theta))^2\right) d\theta$$

(decomposition) $\qquad\qquad$ (1)

$$x'(t) = C \sum_k y'_k(t) \, \exp\left(-j\omega_k t\right)$$

(reconstruction)

where the center frequencies $\omega_k$ are spaced on a logarithmic scale. The constant $\alpha$ sets the relative width of the frequency bins in the decomposition, and can be adjusted (together with $C$) alter the shape of the wavelet kernel. Successive decomposition and reconstruction transforms yield an approximate identity operation; it cannot be exact as no continuous orthonormal basis function exists for the CWT [Kais94].

## 2 Architecture

The above operations are implemented in [Edwa93] using two demodulator systems per channel, one for the real component of (1), and another for the imaginary component, 90° out of phase with the first. Each takes the form of a sinusoidal modulator oscillating at the channel center frequency, followed by a Gaussian-shaped lowpass filter, as shown in Figure 1 (a). This arrangement requires a precise analog sine wave generator and an accurate linear analog multiplier. In the present implementation, we circumvent *both* requirements by using an oversampled binary representation of the modulation reference signal.

### 2.1 Multiplexing vs. Multiplying

Multiplication of an analog signal $x(t)$ with a binary ($\pm 1$) sequence is naturally implemented with high precision using a multiplexer, which alternates between presenting either the input or its inverse $-x(t)$ to the output. This principle is applied to simplify harmonic modulation, and is illustrated in Figure 1 (b). The multiplier has been replaced by an analog inverter followed by a multiplexer, where the multiplexer is controlled by an oversampled binary periodic sequence representing the sine wave reference. The oversampled binary sequence is chosen to approximate the analog sine wave as closely as possible, disregarding components at high frequency which are removed by the subsequent lowpass filter. The assumption made is that no high frequency components are present in the input signal

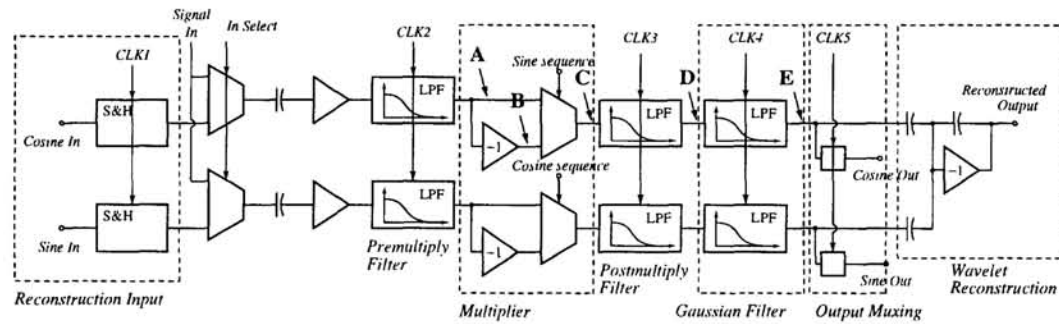

Figure 2: *Block diagram of a single channel in the wavelet processor, showing test points* **A** *through* **E**.

under modulation, which otherwise would convolve with corresponding high frequency components in the binary sequence to produce low frequency distortion components at the output. To that purpose, an additional lowpass filter is added in front of the multiplexer. Residual low-frequency distortion at the output is minimized by maximizing roll-off of the filters, placing proper constraints on their cutoff frequencies, and optimally choosing the bit sequence in the oversampled reference [Edwa95]. Clearly, the signal accuracy that can be achieved improves as the length $N$ of the sequence is extended. Constraints on the length $N$ are given by the implied overhead in required signal bandwidth, power dissipation, and complexity of implementation.

### 2.2   Wavelet Gaussian Function

The reason for choosing a Gaussian kernel in (1) is to ensure optimal support in both time and frequency [Gros89]. A key requirement in implementing the Gaussian filter is linear phase, to avoid spectral distortion due to non-uniform group delays. A worry-free architecture would be an analog FIR filter; however the number of taps required to accommodate the narrow bandwidth required would be prohibitively large for our purpose. Instead, we approximate a Gaussian filter by cascading several first-order lowpass filters. From probabilistic arguments, the obtained lowpass filter approximates a Gaussian filter increasingly well as the number of stages increases [Edwa93].

## 3   Implementation

Two sections of a wavelet processor, each containing 8 parallel channels, were integrated onto a single 4 mm × 6 mm die in 2 μm CMOS technology. Both sections can be configured to perform wavelet decomposition as well as reconstruction. The block diagram for one of the channels is shown in Figure 2. In addition, a separate test chip was designed which performs one channel of the wavelet function. Test points were made available at various points for either input or output, as indicated in boldface capitals, **A** through **E**, in Figure 2.

Each channel performs complex harmonic modulation and Gaussian lowpass filtering, as defined above. At the front end of the chip is a sample-and-hold section to sample time-multiplexed wavelet signals for reconstruction. In cases of both signal decomposition and reconstruction, each channel removes the input DC component removed, filters the result through the premultiplication lowpass (PML) filter, inverts the result, and passes both non-inverted and inverted signals onto the multiplexer. The multiplexer output is passed through a postmultiplication lowpass filter (PML, same architecture) to remove high frequency components of the oversampled sequence, and then passed through the Gaussian-shaped lowpass filter. The cutoff frequencies of all filters are controlled by the clock rates

(CLK1 to CLK4 in Figure 2). The remainder of the system is for reconstruction and for time-multiplexing the output.

### 3.1 Multiplier

The multiplier is implemented by use of the above multiplexing scheme, driven by an oversampled binary sequence representing a sine wave. The sequence we used was 256 samples in length, created from a 64-sample base sequence by reversal and inversion. The sequence length of 256 generates a modulator wave of 4 kHz (useful for speech applications) from a clock of about 1 MHz.

We derived a sequence which, after postfiltering through a 3rd-order lowpass filter of the form of the PML prefilter (see below), produces a sine wave in which all harmonics are more than 60 dB down from the primary [Edwa95]. The optimized 64-bit base sequence consists of 11 zeros and 53 ones, allowing a very simple implementation in which an address decoder decodes the "zero" bits. The binary sequence is shown in Figure 4. The magnitude of the prime harmonic of the sequence is approximately 1.02, within 2% of unity.

The process of reversing and inverting the sequence is simplified by using a gray code counter to produce the addresses for the sequence, with only a small amount of combinatorial logic needed to achieve the desired result [Edwa95]. It is also straightforward to generate the addresses for the cosine channel, which is 90° out of phase with the original.

### 3.2 Linear Filtering

All filters used are implemented as linear cascades of first-order, single-pole filter sections. The number of first-order sections for the PML filters is 3. The number of sections for the "Gaussian" filter is 8, producing a suitable approximation to a Gaussian filter response for all frequencies of interest (Figure 5).

Figure 3 shows one first-order lowpass section of the filters as implemented. This standard

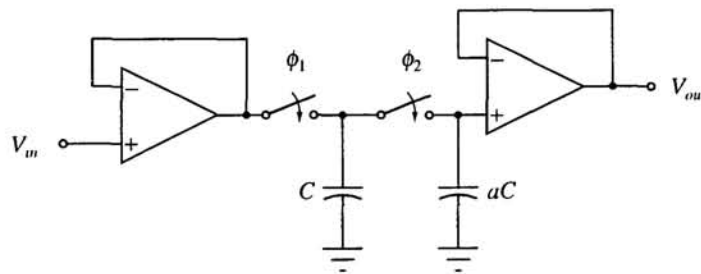

Figure 3: Single discrete-time lowpass filter section.

switched-capacitor circuit implements a transfer function containing a single pole, approximately located in the Laplace domain at $s = f_s / a$ for large values of the parameter $a$, with $f_s$ being the sampling frequency. The value for this parameter $a$ is fixed at the design stage as the ratio of two capacitors in Figure 3, and was set to be 15 for the The PML filters and 12 for the Gaussian filters.

## 4 Measured Results

### 4.1 Sine wave modulator

We tested the accuracy of the sine wave modulation signal by applying two constant voltages at test points **A** and **B**, such that the sine wave modulation signal is effectively multiplied

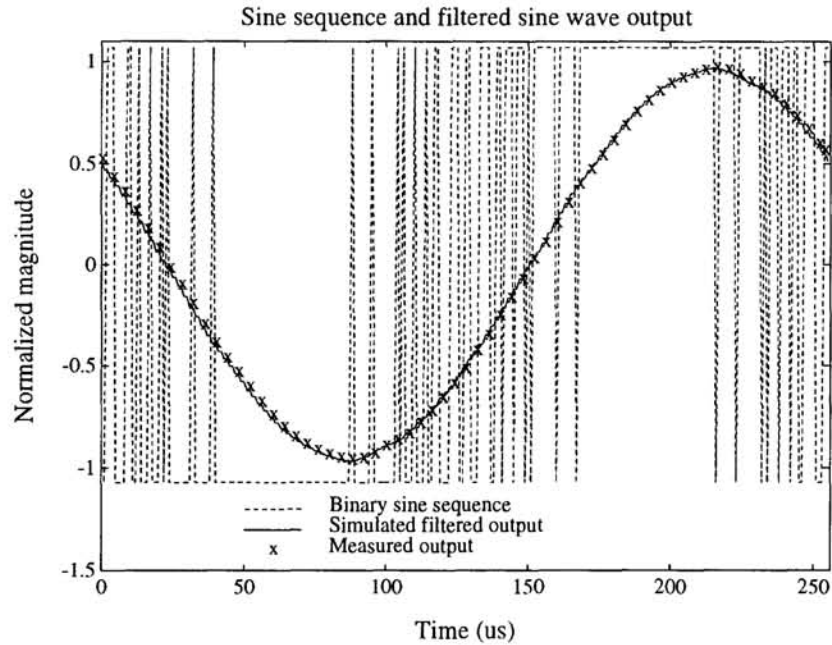

Figure 4: *Filtered sine wave output.*

by a constant. The output of the multiplier is filtered and the output taken at test point **D**, before the Gaussian filter. Figure 4 shows the (idealized) multiplexer output at test point **C**, which accurately creates the desired binary sequence. Figure 4 also shows the measured sine wave after filtering with the PML filter and the expected output from the simulation model, using a deviating value of 8.0 for the capacitor ratio $a$, as justified below. FFT analysis of Figure 4 has shown that the resulting sine wave has all harmonics below about $-49\,dB$. This is in good agreement with the simulation model, provided a correction is made for the value of the capacitor ratio $a$ to account for fringe and (large) parasitic capacitances. The best fit for the measured data from the postmultiplication filter is $a = 8.0$, compared to the desired value of $a = 15.0$. The transform of the simulated output shown in the figure takes into account the smaller value of $a$. Because the postmultiplication filter is followed by the Gaussian filter, the bandwidth of the output can be directly controlled by proper clocking of the Gaussian filter, so the distortion in the sine wave is ultimately much smaller than that measured at the output of the postmultiplication filter.

### 4.2 Gaussian filter

The Gaussian filter was tested by applying a signal at test point **D** and measuring the response at test point **E**. Figure 5 shows the response of the Gaussian filter as compared to expected responses. There are two sets of curves, one for a filter clocked at 64 kHz, and the other clocked at 128 kHz; these curves are normalized by plotting time relative to the clock frequency $f_s$. The solid line indicates the best match for an 8th-order lowpass filter, using the capacitor ratio, $a$, as a fitting parameter. The best-fit value of $a$ is approximately 6.8. This is again much lower than the capacitor area ratio of 12 on the chip. The dotted line is the response of the ideal Gaussian characteristic $\exp\left(-\omega^2/(2a\omega_k^2)\right)$ approximated by the cascade of first-order sections with capacitor ratio $a$.

Figure 5 (b) shows the measured phase response of the Gaussian filter for the 128 kHz clock. The phase response is approximately linear throughout the passband region.

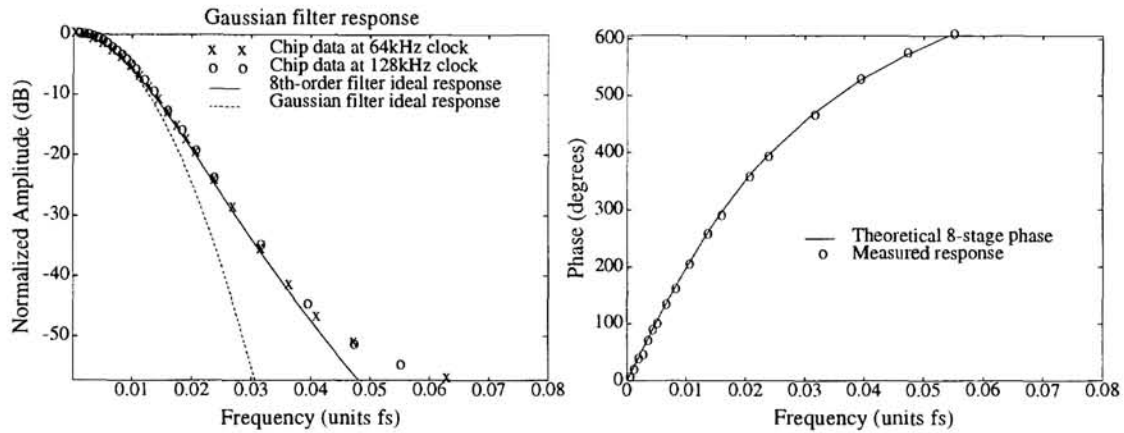

Figure 5: *Gaussian filter transfer functions: theoretical and actual. (a) Relative amplitude; (b) Phase.*

## 4.3   Wavelet decomposition

Figure 6 shows the test chip performing a wavelet transform on a simple sinusoidal input, illustrating the effects of (oversampled) sinusoidal modulation followed by lowpass filtering through the Gaussian filter. The chip multiplier system is clocked at 500 kHz. The input wave is approximately 3.1 kHz, close to the center frequency of the modulator signal, which is the clock rate divided by 128, or about 3.9 kHz (a typical value for the highest-frequency channel in an auditory application). The top trace in the figure shows the filtered and inverted input, taken from test point **B**. The middle trace shows the output of the multiplexer (test point **C**), wherein the output is multiplexed between the signal and its inverse. The bottom trace is taken from the system output (labeled *Cosine Out* in Figure 2) and shows the demodulated signal of frequency 800 Hz (= 3.9 kHz - 3.1 kHz). Not shown is the cosine output, which is 90° out of phase with the one shown. This demonstrates the proper operation of complex demodulation in a single channel configured for wavelet decomposition. In addition, we have tested the full 16-channel chip decomposition, and all individual parts function properly. The total power consumption of the 16-channel wavelet chip was measured to be less than 50 mW, of which a large fraction can be attributed to external interfacing and buffering circuitry at the periphery of the chip.

## 5   Conclusions

We have demonstrated the full functionality of an analog chip performing the continuous wavelet transform (decomposition). The chip is based on mixed analog/digital signal processing principles, and uses a demodulation scheme which is accurately implemented using oversampling methods. Advantages of the architecture used in the chip are an increased dynamic range and a precise control over lateral synchronization of wavelet components. An additional advantage inherent to the modulation scheme used is the potential to tune the channel bandwidths over a wide range, down to unusually narrow bands, since the cutoff frequency of the Gaussian filter and the center frequency of the modulator are independently adjustable and precisely controllable parameters.

## References

G. Kaiser, *A Friendly Guide to Wavelets*, Boston, MA: Birkhäuser, 1994.

T. Edwards and M. Godfrey, "An Analog Wavelet Transform Chip," *IEEE Int'l Conf. on*

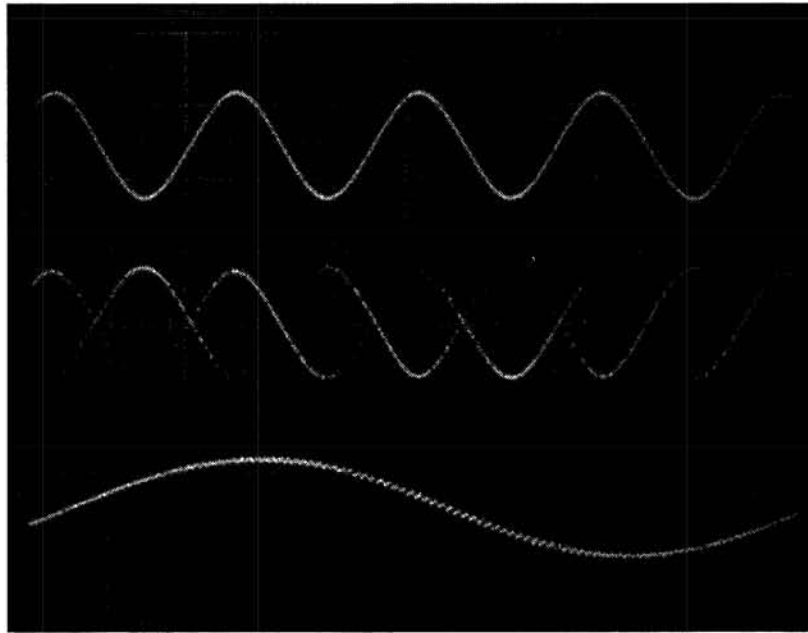

Figure 6: *Scope trace of the wavelet transform: filtered input (top), multiplexed signal (middle), and wavelet output (bottom).*

*Neural Networks*, vol. **III**, 1993, pp. 1247–1251.

T. Edwards and G. Cauwenberghs, "Oversampling Architecture for Analog Harmonic Modulation," to appear in *Electronics Letters*, 1996.

A. Grossmann, R. Kronland-Martinet, and J. Morlet, "Reading and understanding continuous wavelet transforms," *Wavelets: Time-Frequency Methods and Phase Space.* Springer-Verlag, 1989, pp. 2–20.

W. Liu, A.G. Andreou, and M.G. Goldstein, "Voiced-Speech Representation by an Analog Silicon Model of the Auditory Periphery," *IEEE T. Neural Networks*, vol. **3** (3), pp 477–487, 1992.

J. Lin, W.-H. Ki, T. Edwards, and S. Shamma, "Analog VLSI Implementations of Auditory Wavelet Transforms Using Switched-Capacitor Circuits," *IEEE Trans. Circuits and Systems—I*, vol.**41** (9), pp. 572-583, September 1994.

A. Lu and W. Roberts, "A High-Quality Analog Oscillator Using Oversampling D/A Conversion Techniques," *IEEE Trans. Circuits and Systems—II*, vol.**41** (7), pp. 437–444, July 1994.

R.F. Lyon and C.A. Mead, "An Analog Electronic Cochlea," *IEEE Trans. Acoustics, Speech and Signal Proc.,* vol. **36**, pp 1119–1134, 1988.

H.H. Szu, B. Tefler, and S. Kadembe, "Neural Network Adaptive Wavelets for Signal Representation and Classification," *Optical Engineering*, vol. **31** (9), pp. 1907–1916, September 1992.

L. Watts, D.A. Kerns, and R.F. Lyon, "Improved Implementation of the Silicon Cochlea," *IEEE Journal of Solid-State Circuits,* vol. **27** (5), pp 692–700, 1992.